# Memorability of Image Regions

**Aditya Khosla**      **Jianxiong Xiao**      **Antonio Torralba**      **Aude Oliva**
Massachusetts Institute of Technology
`{khosla,xiao,torralba,oliva}@csail.mit.edu`

## Abstract

While long term human visual memory can store a remarkable amount of visual information, it tends to degrade over time. Recent works have shown that image memorability is an intrinsic property of an image that can be reliably estimated using state-of-the-art image features and machine learning algorithms. However, the class of features and image information that is forgotten has not been explored yet. In this work, we propose a probabilistic framework that models how and which local regions from an image may be forgotten using a data-driven approach that combines local and global images features. The model automatically discovers memorability maps of individual images without any human annotation. We incorporate multiple image region attributes in our algorithm, leading to improved memorability prediction of images as compared to previous works.

## 1   Introduction

Human long-term memory can store a remarkable amount of visual information and remember thousands of different pictures even after seeing each of them only once [25, 1]. However, it appears to be the fate of visual memories that they degrade [13, 30]. While most of the work in visual cognition has examined how people forget for general classes of visual or verbal stimuli [30], little work has looked at which image information is forgotten and which is retained. Does all visual information fade alike? Are there some features, image regions or objects that are forgotten more easily than others? Inspired by work in visual cognition showing that humans selectively forget some objects and regions from an image while retaining others [22], we propose a novel probabilistic framework for modeling image memorability, based on the fading of local image information.

Recent work on image memorability [6, 7, 12] has shown that there are large differences between the memorabilities of different images, and these differences are consistent across context and observers, suggesting that memory differences are intrinsic to the images themselves. Using machine learning tools such as support vector regression and a fully annotated dataset of images with human memorability scores, Isola et al [7] show that an automatic image ranking algorithm matches individual image memory scores quite well: with dynamic scenes with people interacting as most memorable, static indoor environments and human-scale objects as somewhat less memorable, and outdoor vistas as forgettable. In addition, using manual annotation, Isola et al. quantified the contribution of segmented regions to the image memorability score, creating a memorability map for each individual image that identifies objects that are correlated with high or low memorability scores. However, this previous work did not attempt to discover in an automatic fashion which part of the image is memorable and which regions are forgettable.

In this paper, we introduce a novel framework for predicting image memorability that is able to account for how memorability of image regions and different types of features fade over time, offering memorability maps that are more interpretable than [7]. The current work offers three original contributions: (1) a probabilistic model that simulates the forgetting local image regions, (2) the automatic discovery of memorability maps of individual images that reveal which regions are memorable/forgettable, and (3) an improved overall image memorability prediction from [7], using an automatic, data-driven approach combining local and global images features.

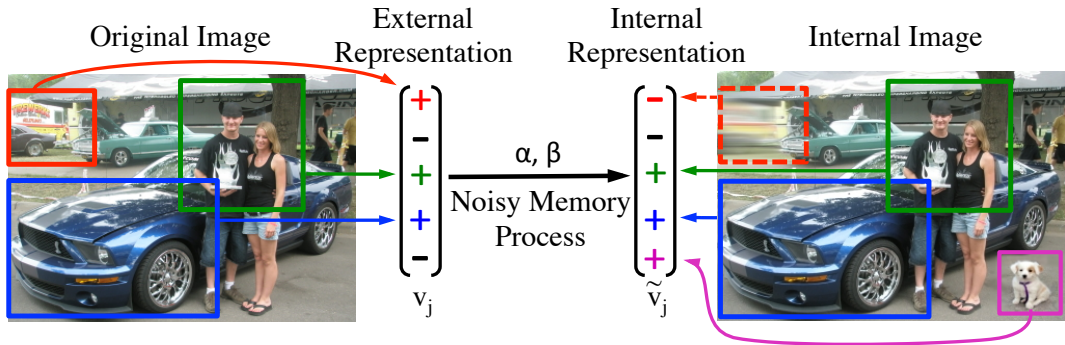

Figure 1: Overview of our probabilistic framework. This figure illustrates a possible external or 'observed' representation of an image. The conversion to an internal representation in memory can be thought of as a noisy process where some elements of the image are changed probabilistically as described by $\alpha$ and $\beta$ (Sec. 3.1). The image on the right illustrates a possible internal representation: the **green** and **blue** regions remain unchanged, while the **red** region is forgotten and the **pink** region is hallucinated. Note that the internal representation cannot be observed and is only shown here for illustrating the framework.

## 2  Related work

Large scale visual memory experiments [26, 25, 1, 13, 14, 28] have shown that humans can remember specific images they have seen among thousands of images, hours to days later, even after being exposed to each picture only once. In addition, humans seem to have a massive capacity in long term memory to store specific details about these images, like remembering whether the glass of orange juice they saw thousands of images earlier was full or half full [1] or which specific door picture they saw after being exposed to hundreds of pictures of doors [28].

However, not all images are equally memorable as shown by the Memory Game experiment described in [7, 12], and importantly, not all kinds of local information are equally retained from an image: on average, observers will more likely remember visual details attached to objects that have a specific semantic label or a distinctive interpretation (for example observers will remember different types of cars by tagging each car with a different brand name, but would more likely confuse different types of apples, which only differ by their color [14]). This suggests that different features, objects and regions in an image may have themselves different memorability status: indeed, works by Isola et al [7, 6] have shown that different individual features, objects, local regions and attributes are correlated with image that are highly memorable or forgettable. For instance, indoor spaces, pictures containing people, particularly if their face is visible, close up views on objects, animals, are more memorable than buildings, pictures of natural landscapes, and natural surfaces in general (like mountains, grass, field). However, to date, there is no work which has attempted to predict which local information from an image is memorable or forgettable, in an automatic manner.

## 3  Modeling memorability using image regions

We propose to predict memorability using a noisy memory process of encoding images in our memory, illustrated in Fig. 1. In our setting, an image consists of different types of image regions and features. After a delay between the first and second presentation of an image, people are likely to remember some image regions and objects more than others. For example, as shown in [7], people and close up views on objects tend to be more memorable than natural objects and regions of landscapes, suggesting for instance that an image region containing a person is less likely to be forgotten than an image region containing a tree. It is well established that stored visual information decays over time [30, 31, 14], which can be represented in a model by a novel image vector with missing global and local information. We postulate that the farther the stored representation of the image is from its veridical representation, the less likely it is to be remembered.

Here, we propose to model this noisy memorability process in a probabilistic framework. We assume that the representation of an image is composed of image regions where different regions of an

image correspond to different sets of objects. These regions have different probabilities of being forgotten and some regions have a probability of being imagined or hallucinated. We postulate that the likelihood of an image to be remembered depends on the distance between the initial image representation and its internal degraded version. An image with a larger distance to the internal representation is more likely to be forgotten, thereby the image should have a lower memorability score. In our algorithm, we model this probabilistic process and show its effectiveness at predicting image memorability and at producing interpretable memorability maps.

## 3.1 Formulation

Given some image $I_j$, we define its representation $v_j$ and $\tilde{v}_j$ as the external and internal representation of the image respectively. The external representation refers to the original image which is observed, while internal representation refers to the noisy representation of the same image that is stored in the observer's memory. Assume that there are $N$ types of regions or objects an image can contain. We define $v_j \in \{0,1\}^N$ as a binary vector of size $N$ containing a 1 at index $n$ when the corresponding region is present in image $I_j$ and 0 otherwise. Similarly, the internal representation consists of the same set of region types, but has different presence and absence values as memory is noisy.

In this setting, one of two things can happen when the external representation of an image is observed: (1) An image region that was shown is forgotten i.e. $\tilde{v}_j(i) = 0$ when $v_j(i) = 1$, where $v_j(i)$ refers to the $i^{th}$ element of $v_j$, or (2) An image region is hallucinated i.e. an image region that did not exist in the image is believed to be present. We expect this to happen with different probabilities for different types of image regions. Therefore, we define two probability vectors $\vec{\alpha}, \vec{\beta} \in [0,1]^N$, where $\alpha_i$ corresponds to the probability of region type $i$ being forgotten while $\beta_i$ corresponds to the probability of hallucinating a region of type $i$.

Using this representation, we define the distance between the internal and external representation as $D_j = D(v_j, \tilde{v}_j) = ||v_j - \tilde{v}_j||_1$. $D_j$ is inversely proportional to the memorability score of an image $s_j$; the higher the distance of an image in the brain from its true representation, the less likely it is to be remembered, i.e. when $D$ increases, $s$ decreases. Thus, we can compute the expected distance $E(D_j|v_j)$ of an image as:

$$E(D_j|v_j) = \sum_{i=1}^{N} \alpha_i^{v_j(i)} * \beta_i^{1-v_j(i)} = v_j^T \vec{\alpha} + (\neg v_j)^T \vec{\beta} \tag{1}$$

This represents the expected number of modifications in $v$ from 1 to 0 ($\alpha$) or from 0 to 1 ($\beta$). Thus, over all images, we can define the expected distance $E(D|v)$ as

$$E(D|v) = \begin{pmatrix} -\ v_1^T\ - & -\neg v_1^T\ - \\ \vdots & \vdots \\ -\ v_M^T\ - & -\neg v_M^T\ - \end{pmatrix} \cdot \begin{pmatrix} \vec{\alpha} \\ \vec{\beta} \end{pmatrix} \propto_{rank} -\vec{s} \tag{2}$$

where $\alpha_i, \beta_i \in [0,1]$ and $\propto_{rank}$ represents that the proportionality is only related to the relative ranking of the image memorability scores, and $M$ is the total number of images. We do not explicitly predict a memorability score, rather the ranking of scores between images.

The above equation represents a typical ordinal rank regression setting with additional constraints on the learning parameters $\vec{\alpha}$ and $\vec{\beta}$. Since we are only interested in the rank, we can rescale the learned parameters to lie between $[0,1]$, allowing us to use standard solvers such as SVM-Rank [9]. We note that $\vec{\beta}$ cannot be uniquely determined when considering ranking of images alone, and thus we focus our attention on $\vec{\alpha}$ for the rest of this paper.

**Implementation details:** To generate the region types automatically, we randomly sample rectangular regions of arbitrary width and height from the training images. The regions can be overlapping with each other. For each region, we compute a particular feature (described in Sec. 4.2), ensuring the same dimension for all regions of different shapes and sizes (using Bag-of-Words like representations). Then we perform k-means clustering to learn the dictionary of region types as cluster centroids. The region type is determined by the closest cluster centroid. This method allows us to

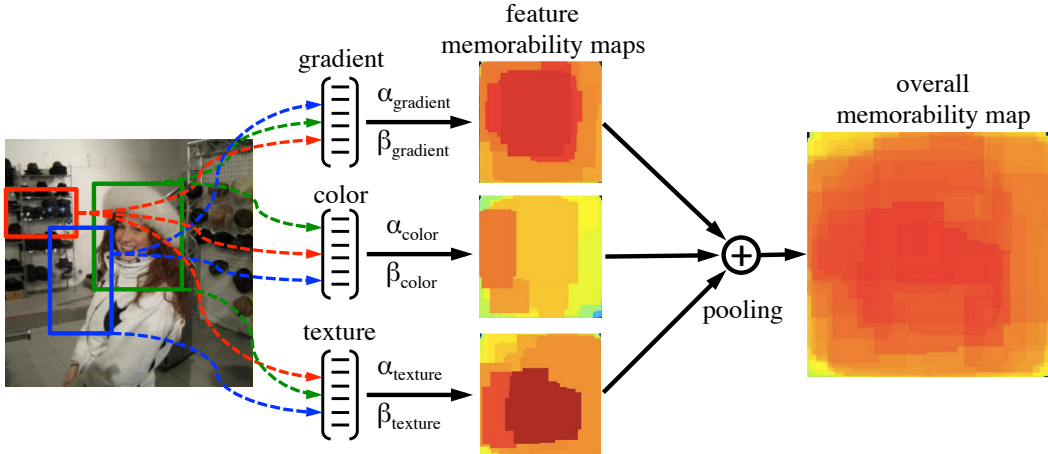

Figure 2: Illustration of multiple feature integration. Refer to Sec. 3.2 for details.

bypass the need for human annotation as done in [7]. The details of the dictionary size and feature types used are provided in Sec. 4. As we sample overlapping regions, we only encode the presence of a region type by $1$ or $0$. There may be more than one sampled region that corresponds to a particular region type.

We evaluate our algorithm on test images by applying a similar method as that on the train images. In this case, we assume the dictionary of region types is given, and we simply assign the randomly sampled image regions to region types, and use the learned parameters $(\vec{\alpha}, \vec{\beta})$ to compute a score.

## 3.2  Multiple feature integration

We incorporate multiple attributes of each region type such as color, texture and gradient in the form of image features into our algorithm. Our method is illustrated in Fig. 2. For each attribute, we learn a separate dictionary of region types. An image region is encoded using each feature dictionary independently, and the $\vec{\alpha}, \vec{\beta}$ parameters are learned jointly in our learning algorithm. Subsequently, we use each set of $\vec{\alpha}$ and $\vec{\beta}$ for individual features to construct memorability maps that are later combined using weighted pooling[1] to produce an overall memorability map as shown in Fig. 2. We demonstrate experimentally (Sec. 4) that multiple feature integration helps to improve both the memorability score prediction and produce visually more consistent memorability maps.

## 4  Experiments

In this section, we describe the experimental setup and dataset used (Sec. 4.1), provide details about the region attributes used in our experiments (Sec. 4.2) and describe the experimental results on the image memorability dataset (Sec. 4.3). Experimental results show that our method outperforms state-of-the-art methods on this dataset while providing automatic memorability maps of images that compare favorably to when ground truth segmentation is used.

### 4.1  Setup

**Dataset:** We use the dataset proposed by Isola et al. [7] consisting of 2222 images from the SUN dataset [32]. The images are fully annotated with segmented object regions and randomly sampled from different scene categories. The images are cropped and resized to $256 * 256$ and a memorability score corresponding to each image is provided. The memorability score is defined as the percentage of correct detections by participants in their study.

**Performance evaluation:** The performance is evaluated using Spearman's rank correlation($\rho$). We evaluate our performance on 25 different training/testing splits of the data (same splits as [7]) with

an equal number of images for training and testing (1111). The train splits are scored by one half of the participants and the test splits are scored by the other half of the participants with a human consistency of $\rho = 0.75$. This can be thought of as an upper bound in the performance of automatic methods.

**Algorithmic details:** We sample 2000 patches per image with size $0.2 * 0.2$ to $0.7 * 0.7$ with random aspect ratios in normalized image coordinates. To speed up convergence of SVM-Rank, we do not include rank constraints for memorability scores that lie within 0.001 of each other. We find that this does not affect the performance significantly. The hyperparameter of the SVM-Rank algorithm is set using 5-fold cross-validation.

## 4.2 Image region attributes

Our goal is to choose various features as attributes that human likely use to represent image regions. In this work, we consider six common attributes, namely gradient, color, texture, shape, saliency and semantic meaning of the images. The attributes are extracted for each region and assigned to a region type as described in Sec. 3.2 with a dictionary size of 1024 for each feature. For each of the attributes, we describe our motivation and the method used for extraction.

**Gradient:** In human vision system, much evidence suggests that retinal ganglion cells and receptive fields of cells in the visual cortex V1 are essentially gradient-based features. Furthermore, recent success of many computer vision algorithms [2, 4] also demonstrated the power of such features. In this work, we use the powerful Histogram of Oriented Gradients (HOG) features for our task. We densely sample HOG [2] with a cell size of 2x2 at a grid spacing of 4 and learn a dictionary of size 256. The descriptors for a given image region are max-pooled at 2 spatial pyramid levels[15] using Locality-Constrained Linear Coding (LLC) [29].

**Color:** Color is an important part of human vision. Color usually has large variations caused by changes in illumination, shadows, etc, and these variations make the task of robust color description difficult. Isola et al. [7] show that simple image color features, such as mean hue, saturation and intensity, only exhibits very weak correlation with memorability. In contrast to this, color has been shown to yield excellent results in combination with shape features for image classification [11]. Furthermore, many studies show that color names are actually linguistic labels that humans assign to color spectrum space. In this paper, we use the color names feature [27] to better exploit the color information. We densely sample the feature at multiple scales (12, 16, 24 and 32) with a grid spacing of 4. Then we learn a dictionary of size 100 and apply LLC at 2-level spatial pyramid to obtain the color descriptor for each region.

**Texture:** We interact with a variety of materials on a daily basis and we constantly assess their texture properties by visual means and tactile touch. To encode visual texture perception information, we make use of the popular texture features – Local Binary Pattern [21] (LBP). We use a 2-level spatial pyramid of non-uniform LBP descriptor.

**Saliency:** Image saliency is a biologically inspired model to capture the regions that attract more visual attention and fixation focus [8]. Inspired by this, we extract a saliency value for each pixel using natural statistics [10]. Then we perform average pooling at 3-level spatial pyramid to obtain the descriptor for each region.

**Shape:** Humans constantly use geometric patterns to determine the similarity between visual entities, and the layout of shapes is directly relevant to mid level representations of the image. We denote shape as a histogram of local Self-Similarity geometric pattens (SSIM [23]). We densely sample the SSIM descriptor with a grid spacing of 4 and learn a dictionary of sie 256. The descriptors for a given image region are max-pooled at 2 spatial pyramid levels using LLC.

**Semantic:** High-level semantic meaning contained in images has been shown to be strongly correlated to image memorability [7], where manual annotation of object labels lead to great performance in predicting image memorability. Here, our goal is to design a fully automatic approach to predict image memorability, while still exploiting the semantic information. Thus, we use the automatic Object Bank [17] feature to model the presence/absence of various objects in the images. We reduce the feature dimension by using simple max pooling instead of spatial pyramid pooling.

Table 1: Images are sorted into sets according to predictions made on the basis of a variety of features (denoted by column headings). Average measured memorabilities are reported for each set. e.g. The Top 20 row reports average measured memorability of the 20 highest predicted images. $\rho$ is the Spearman rank correlation between predictions and measurements.

|  | Multiple global features [7] | Our Global | Our Local | Our Full Model |
|---|---|---|---|---|
| Top 20 | 83% | 84% | 83% | 85% |
| Top 100 | 80% | 80% | 80% | 81% |
| Bottom 100 | 57% | 56% | 57% | 55% |
| Bottom 20 | 55% | 53% | 54% | 52% |
| $\rho$ | 0.46 | 0.48 | 0.45 | **0.50** |

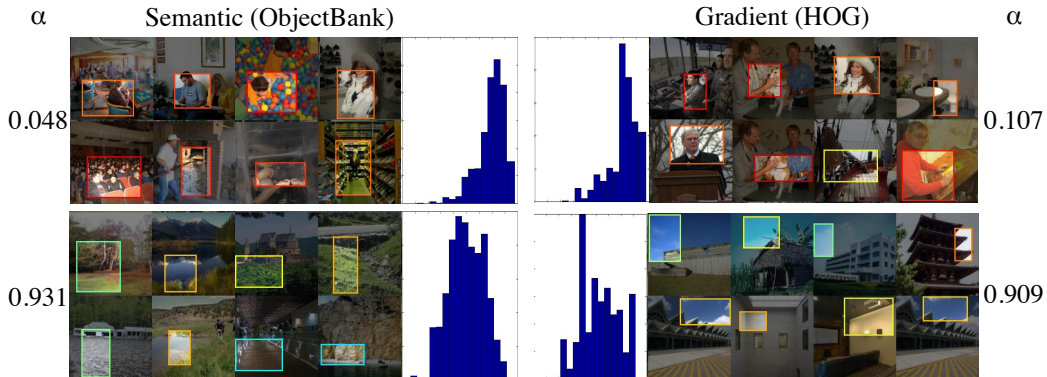

Figure 3: Visualization of region types and corresponding $\alpha$ learned by our algorithm for gradient and semantic features. The histograms represent the distribution of memorability scores corresponding to the particular region type. We observe that high-scoring images tend to have a small value of $\alpha$ while low scoring regions have a high value. This corresponds well with the proposed framework. The color of the bounding boxes corresponds to the memorability score of the image shown (using a jet color scheme).

## 4.3 Results

In this section, we evaluate the performance of our model with single and multiple features, and later explore what the model has learned using memorability maps and the ranking of different types of image regions.

**Single + multiple features:** Fig. 6(a) and Tbl. 1 summarize the performance of our algorithm when using single and multiple features. We compare our results with [7], and find that our algorithm outperforms the automatic methods from [7] by 4%, and achieve comparable performance to when ground truth annotation is used. This shows the effectiveness of our method at predicting memorability. Further, we note that our model provides complementary information to global features as it focuses on local image regions, increasing performance by 2% when combined with our global features. We use the same set of attributes described in Sec. 4.2 as global features in our model. The global features are learned independently using SVM-Rank and the predicted score is combined with the predicted scores of our local model in SVM-Rank algorithm. Despite using the same set of features, we are able to obtain performance gain suggesting that our algorithm is effective at capturing local information in the image that was overlooked by the global features.

**Memorability maps:** We obtain memorability maps using max-pooling of the $\alpha$ from different image regions. Fig. 4 shows the memorability maps obtained when using different features and the overall memorability map when combining multiple features. Despite using no annotation, the learned maps are similar to those obtained using ground truth objects and segments. From the images shown, we observe that there is no single attribute that is always effective at producing memorability maps, but the combination of the attributes leads to a significantly improved version. We show additional results in Fig. 5.

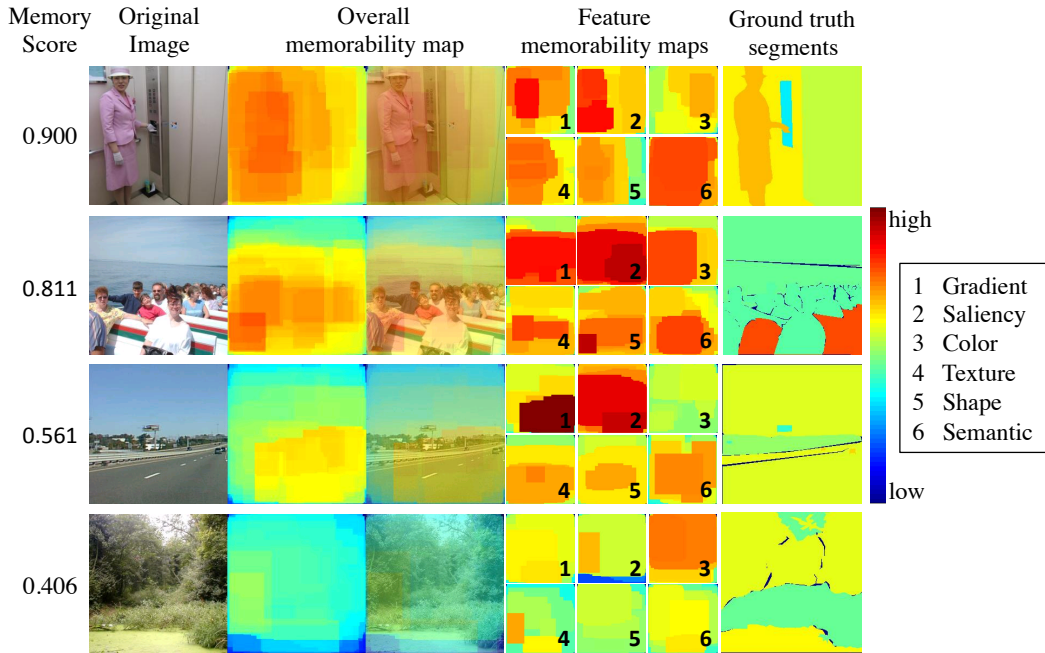

Figure 4: Visualization of the memorability maps obtained using different features, and the overall memorability map. Additionally, we also include the memorability map obtained when using ground truth segmentation on the right. We observe that it resembles our automatically generated maps.

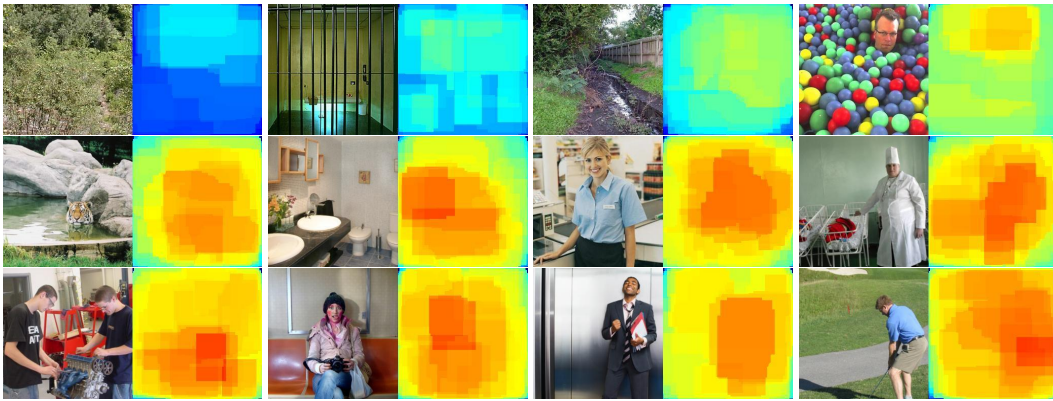

Figure 5: Additional examples of memorability maps generated by our algorithm.

**Image region types:** In Fig. 3, we rank the image region types by their $\alpha$ value and visualize the regions for the corresponding region type when $\alpha$ is close to $0$ or $1$. We observe that the region types are consistent with our intuition of what is memorable from [7]. People often exist in image regions with low $\alpha$ (i.e. low probability of being forgotten) while natural scenes and plain backgrounds are observed in high $\alpha$.

Further, we analyze the image region types by computing the standard deviation of the memorability scores of the image regions that correspond to the particular type. Fig. 6(b) and 6(c) show the results. The results are encouraging as regions that have high standard deviation tend to have a value of $\alpha$ close to $0.5$, which means they are not very informative for prediction. The same behavior is observed for multiple feature types, and we find that the overall performance for individual features (shown in Fig. 6(a)) corresponds well with the distance of the peaks in Fig. 6(b) from $\alpha = 0.5$. This suggests that our algorithm is effective at learning the regions with high and low probability of being forgotten as proposed in our framework.

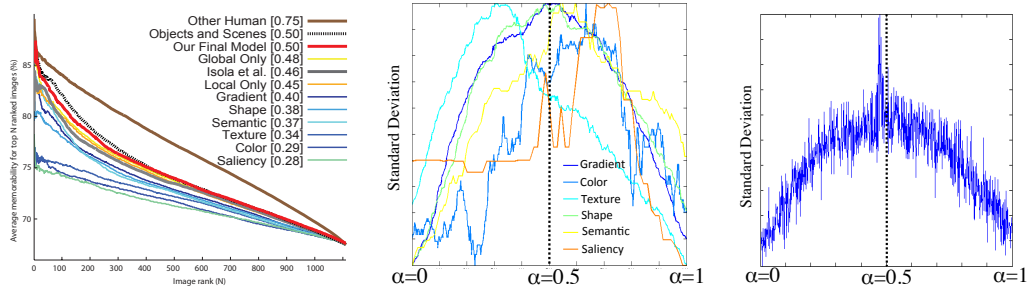

(a) Comparison of results averaged across the 25 splits. Images are ranked by predicted memorability and plotted against the cumulative average of measured memorability scores.

(b) Standard deviation of memorability score of all region types averaged across the 25 splits for all features, sorted by $\alpha$. Graphs are smoothed using a median filter.

(c) Standard deviation of region types for Gradient feature averaged across the 25 splits. No smoothing is applied in this case.

Figure 6: Plot of various results and analysis of our method. Fig. 6(b) and Fig. 6(c) are explained in greater detail in Sec. 4.3

## 5 Conclusion

With the emergence of large scale photo collections and growing demands in storing, organizing, interpreting, and summarizing large amount of digital information, it becomes essential to be able to automatically annotate images on various novel dimensions that are interpretable to human users. Recently, learning algorithms have been proposed to automatically interpret whether an image is aesthetically pleasant or not [20, 3], memorable or forgettable [7, 6], and the role that other high level photographic properties plays in image interpretation (photo quality [19], attractiveness [16], composition [5, 18], and object importance [24]). Here, we propose a novel probabilistic framework for automatically constructing memorability maps, discovering regions in the image that are more likely to be memorable or forgettable by human observers. We demonstrate an effective yet interpretable framework to model the process of forgetting. Future development of such automatic algorithms of image memorability could have many exciting and far-reaching applications in computer science, graphics, media, designs, gaming and entertainment industries in general.

## Acknowledgements

We thank Phillip Isola and the reviewers for helpful discussions. This work is funded by NSF grant (1016862) to A.O, Google research awards to A.O and A.T, ONR MURI N000141010933 and NSF Career Award (0747120) to A.T. J.X. is supported by Google U.S./Canada Ph.D. Fellowship in Computer Vision.

## Footnotes

[1]We weight the importance of individual features by summing the $\vec{\alpha}$ corresponding to the particular feature.

## References

[1] T. F. Brady, T. Konkle, G. A. Alvarez, and A. Oliva. Visual long-term memory has a massive storage capacity for object details. *PNAS*, pages 14325–14329, 2008.

[2] N. Dalal and B. Triggs. Histograms of oriented gradients for human detection. In *CVPR*, volume 1, pages 886–893. IEEE, 2005.

[3] S. Dhar, V. Ordonez, and T.L. Berg. High level describable attributes for predicting aesthetics and interestingness. In *CVPR*, pages 1657–1664. IEEE, 2011.

[4] P.F. Felzenszwalb, R.B. Girshick, D. McAllester, and D. Ramanan. Object detection with discriminatively trained part-based models. *TPAMI*, 2010.

[5] B. Gooch, E. Reinhard, C. Moulding, and P. Shirley. Artistic composition for image creation. In *Rendering Techniques 2001: Proceedings of the Eurographics Workshop in London, United Kingdom, June 25-27, 2001*, page 83. Springer Verlag Wien, 2001.

[6] P. Isola, D. Parikh, A. Torralba, and A. Oliva. Understanding the intrinsic memorability of images. In *Advances in Neural Information Processing Systems (NIPS)*, 2011.

[7] P. Isola, J. Xiao, A. Torralba, and A. Oliva. What makes an image memorable? In *IEEE Conference on Computer Vision and Pattern Recognition (CVPR)*, pages 145–152, 2011.

[8] L. Itti and C. Koch. A saliency-based search mechanism for overt and covert shifts of visual attention. *Vision Research*, 40:1489–1506, 2000.

[9] T. Joachims. Training linear SVMs in linear time. In *ACM SIGKDD*, pages 217–226, 2006.

[10] C. Kanan, M.H. Tong, L. Zhang, and G.W. Cottrell. Sun: Top-down saliency using natural statistics. *Visual Cognition*, 17(6-7):979–1003, 2009.

[11] F. S. Khan, J. van de Weijer, A. D. Bagdanov, and M. Vanrell. Portmanteau vocabularies for multi-cue image representation. In *NIPS*, Granada, Spain, 2011.

[12] A. Khosla*, J. Xiao*, P. Isola, A. Torralba, and A. Oliva. Image memorability and visual inception. In *SIGGRAPH Asia*, 2012. * indicates equal contribution.

[13] T. Konkle, T.F. Brady, G.A Alvarez, and A. Oliva. Conceptual distinctiveness supports detailed visual long-term memory for real-world objects. *Journal of Experimental Psychology*, (139):558–578, 3 2010.

[14] T. Konkle, T.F. Brady, G.A. Alvarez, and A. Oliva. Scene memory is more detailed than you think: the role of categories in visual long-term memory. *Psychological Science*, (21):1551–1556, 11 2010.

[15] S. Lazebnik, C. Schmid, and J. Ponce. Beyond bags of features: Spatial pyramid matching for recognizing natural scene categories. In *CVPR*, volume 2, pages 2169–2178. IEEE, 2006.

[16] T. Leyvand, D. Cohen-Or, G. Dror, and D. Lischinski. Data-driven enhancement of facial attractiveness. In *ACM Transactions on Graphics (TOG)*, volume 27, page 38. ACM, 2008.

[17] L.-J. Li, H. Su, E. P. Xing, and L. Fei-Fei. Object bank: A high-level image representation for scene classification & semantic feature sparsification. In *NIPS*, Vancouver, Canada, December 2010.

[18] L. Liu, R. Chen, L. Wolf, and D. Cohen-Or. Optimizing photo composition. In *Computer Graphics Forum*, volume 29, pages 469–478. Wiley Online Library, 2010.

[19] Y. Luo and X. Tang. Photo and video quality evaluation: Focusing on the subject. In *Proceedings of the 10th European Conference on Computer Vision: Part III*, pages 386–399. Springer-Verlag, 2008.

[20] L. Marchesotti, F. Perronnin, D. Larlus, and G. Csurka. Assessing the aesthetic quality of photographs using generic image descriptors. In *Computer Vision (ICCV), 2011 IEEE International Conference on*, pages 1784–1791. IEEE, 2011.

[21] T. Ojala, M. Pietikainen, and T. Maenpaa. Multiresolution gray-scale and rotation invariant texture classification with local binary patterns. *Pattern Analysis and Machine Intelligence*, 24(7):971–987, 2002.

[22] R. A. Rensink, J. K. O'Regan, and J. J. Clark. To See or not to See: The Need for Attention to Perceive Changes in Scenes. *Psychological Science*, 8(5):368–373, September 1997.

[23] E. Shechtman and M. Irani. Matching local self-similarities across images and videos. In *Computer Vision and Pattern Recognition, 2007. CVPR'07. IEEE Conference on*, pages 1–8. Ieee, 2007.

[24] M. Spain and P. Perona. Some objects are more equal than others: Measuring and predicting importance. *Computer Vision–ECCV 2008*, pages 523–536, 2008.

[25] L. Standing. Learning 10000 pictures. *The Quarterly journal of experimental psychology*, 25(2):207–222, 1973.

[26] L. Standing, J. Conezio, and R.N. Haber. Perception and memory for pictures: Single-trial learning of 2500 visual stimuli. *Psychonomic Science; Psychonomic Science*, 1970.

[27] J. Van De Weijer, C. Schmid, and J. Verbeek. Learning color names from real-world images. In *Computer Vision and Pattern Recognition, 2007. CVPR'07. IEEE Conference on*, pages 1–8. IEEE, 2007.

[28] S. Vogt and S. Magnussen. Long-term memory for 400 pictures on a common theme. *Experimental Psychology (formerly Zeitschrift für Experimentelle Psychologie)*, 54(4):298–303, 2007.

[29] J. Wang, J. Yang, K. Yu, F. Lv, T. Huang, and Y. Gong. Locality-constrained linear coding for image classification. In *CVPR*, pages 3360–3367. IEEE, 2010.

[30] J. T. Wixted. The Psychology and Neuroscience of Forgetting. *Annual Review of Psychology*, 55(1), 20040101.

[31] J. T. Wixted and S. K. Carpenter. The Wickelgren Power Law and the Ebbinghaus Savings Function. *Psychological Science*, 18(2):133–134, February 2007.

[32] J. Xiao, J. Hays, K.A. Ehinger, A. Oliva, and A. Torralba. SUN database: Large-scale scene recognition from abbey to zoo. In *CVPR*, pages 3485–3492. IEEE, 2010.

